# A Machine Learning Approach to Predict Chemical Reactions

**Matthew A. Kayala**    **Pierre Baldi**[*]
Institute of Genomics and Bioinformatics
School of Information and Computer Sciences
University of California, Irvine
Irvine, CA 92697
{mkayala,pfbaldi}@ics.uci.edu

## Abstract

Being able to predict the course of arbitrary chemical reactions is essential to the theory and applications of organic chemistry. Previous approaches are not high-throughput, are not generalizable or scalable, or lack sufficient data to be effective. We describe single mechanistic reactions as concerted electron movements from an electron orbital source to an electron orbital sink. We use an existing rule-based expert system to derive a dataset consisting of 2,989 productive mechanistic steps and 6.14 million non-productive mechanistic steps. We then pose identifying productive mechanistic steps as a ranking problem: rank potential orbital interactions such that the top ranked interactions yield the major products. The machine learning implementation follows a two-stage approach, in which we first train atom level reactivity filters to prune $94.0\%$ of non-productive reactions with less than a $0.1\%$ false negative rate. Then, we train an ensemble of ranking models on pairs of interacting orbitals to learn a relative productivity function over single mechanistic reactions in a given system. Without the use of explicit transformation patterns, the ensemble perfectly ranks the productive mechanisms at the top $89.1\%$ of the time, rising to $99.9\%$ of the time when top ranked lists with at most four non-productive reactions are considered. The final system allows multi-step reaction prediction. Furthermore, it is generalizable, making reasonable predictions over reactants and conditions which the rule-based expert system does not handle.

## 1   Introduction

Determining the major products of chemical reactions given the input reactants and conditions is a fundamental problem in organic chemistry. Reaction prediction is a necessary component of retro-synthetic analysis or virtual library generation for drug design[1, 2] and has the potential to increase our understanding of biochemical catalysis and metabolism[3]. There are a broad range of approaches to reaction prediction falling around at least three main poles: physical simulations of transition states using various quantum mechanical and other approximations[4, 5, 6], rule-based expert systems[2, 7, 8, 9, 10, 11], and inductive machine learning methods[12]. However, none of these approaches can successfully emulate the remarkable abilities of a human chemist.

### 1.1   Previous approaches and representations

The very concept of a "reaction" can be ambiguous, as it corresponds to a macroscopic abstraction, hence simplification, of a very complex underlying microscopic reality, ultimately driven by the

---

[*]To whom correspondence should be addressed

laws of quantum and statistical mechanics. However, even for relatively small systems, it is impossible to find exact solutions to the Schrödinger equation. Thus in practice, energies are calculated with varyingly accurate approximations, ranging from ab-initio Hartree-Fock approaches or density functional theory to semi-empirical methods or mechanical force fields[6]. This leads to modeling reactions as minimum energy paths between stable atom configurations on a high-dimensional potential energy surface, where the path through the lowest energy transition state, i.e., saddle point, is the most favorable[4, 5]. By explicitly modeling energies, these approaches can be highly accurate and generalize to a diverse range of chemistries but require careful initialization and are computationally expensive (see [13] for a representative example). This branch of computational chemistry provides invaluable tools for in-depth analysis but is currently not suitable for high-throughput reactivity tasks and is far from being able to recapitulate the knowledge and ability of human experts.

In contrast, most rule-based expert systems for high-throughput reactivity tasks use a much more abstract representation, in the form of general transformations over molecular graphs[2, 7, 8, 9, 10]. Reactions are predicted when a match is found in a library of allowable graph transformations. These general transformations model only net molecular changes for processes that in reality involve a sequence of transition states, as shown in Figure 1. These rule-based approaches suffer from at least four drawbacks: (1) they use a representation that is too high-level, in that an overall transformation obfuscates the underlying physical reality; (2) they require the manual curation of large amounts of expert knowledge; (3) they become unmanageable at larger scales, in that adding a new graph pattern often involves having to update a large proportion of existing transformations with exceptions; and (4) they lack generality, in that particular chemistries must explicitly be encoded to be predicted.

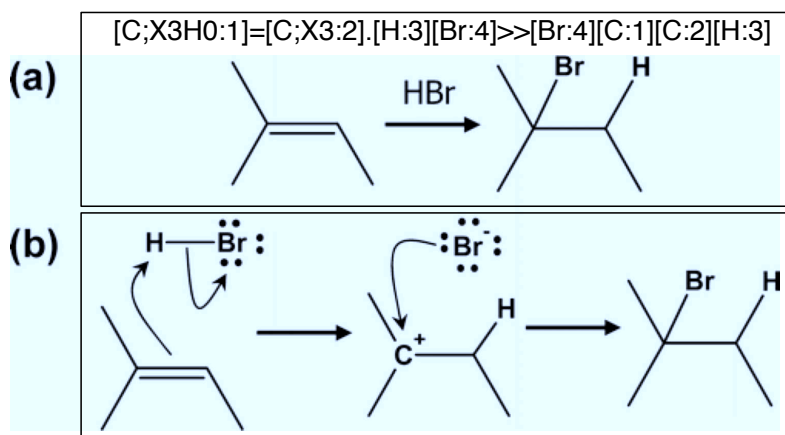

Figure 1: Overall transformation of an alkene (hydrocarbon with double bond) with hydrobromic acid (HBr) and corresponding mechanistic reactions. (a) shows the overall transform as a SMIRKS[14] string pattern and as a graph representation. In a molecular graph, vertices represent atoms, with carbons at unlabeled vertices. The number of edges between two vertices represents bond order. $+/-$ symbols represent formal charge. Standard valences are filled using implicit hydrogens. (b) shows the two mechanistic reactions composing the overall transformation as arrow-pushing diagrams[15, 16]. Dots represent non-bonded (lone pair) electrons, while arrows represent concerted electron movement. In the first step, electrons in the electron-rich carbon-carbon double bond attack the hydrogen and break the electron-poor hydrogen-bromine single bond, producing an anionic bromide (Br−) and a carbocation (C+). In the second step, electrons from the charged, electron-rich bromide attack the electron-poor carbocation, yielding the final alkyl halide.

Somewhere between low-level QM treatment and abstract graph-based overall transformations, one can consider reactions at the mechanistic level. A mechanistic, or elementary, reaction is a concerted electron movement through a single transition state[15, 16]. These mechanistic reactions can be composed to yield overall transformations. For example, Figure 1 shows the overall transformation of an alkene interacting with hydrobromic acid to yield an alkyl halide, along with the two elementary reactions which compose the transformation. A mechanistic reaction is described as an idealized molecular orbital (MO) interaction between an electron source (donor) MO and an electron sink (acceptor) MO. MOs represent regions of the molecule with high (source) or low (sink) electron

density. In general, potential electron sources are composed of lone pairs of electrons and bonds, and potential electron sinks are composed of empty atomic orbitals and bonds. Bonds can act as either a source or a sink depending on the context. Because of space constraints, we cannot fully describe subtle chemical details that must be handled, such as chaining for resonance rearrangement. For details, see texts[15, 16] on mechanisms. Note that by considering all possible pairings of source and sink MOs, this representation allows the exhaustive enumeration of all potential mechanistic reactions over an arbitrary set of molecules.

Recent work by Chen and Baldi[11] introduces a rule-based expert system (Reaction Explorer) in which each rearrangement pattern encompasses an elementary reaction. Here, the elementary reactions represent "productive" mechanistic steps, i.e. those reactions which lead to the overall major products. Thus, elementary reactions which are not the most kinetically favorable, but which eventually lead to the overall thermodynamic transformation product may be considered "productive". This approach is a marked change from previous approaches using overall transformations, but as a rule-based system still suffers from the problems of curation, scale, and generality.

While mechanistic reaction representations are approximations quite far from the Schrödinger equation, we expect them to be closer to the underlying reality and therefore more useful than overall transformations. Furthermore, we expect them also to be easier to predict than overall transformations due to their more elementary nature and mechanistic interpretation. In combination, these arguments suggest that working with mechanistic steps may facilitate the application of statistical machine learning approaches, and take advantage of their capability to generalize. Thus, in this work, reactions are modeled as mechanisms, and for the remainder of the paper, we consider the term "reaction" to denote a single elementary reaction. Furthermore, we consider the problem of reaction prediction to be precisely that of identifying the "productive" reactions over a given set of reactants under particular conditions.

There has been very little work on machine learning approaches to reaction prediction. The sole example is a paper from 1990 on inductively extracting overall transformation patterns from reaction databases[12], a method which was never actually incorporated into a full reaction prediction system. This situation is surprising. Given improvements in both computing power and machine learning methods over the past 20 years, one could imagine a machine learning system that mines reaction information to learn the grammar of chemistry, e.g., in terms of graph grammars[17]. One potential reason behind the lack of progress in this area is the paucity of available data. Chemical publishing is dominated by closed models, making literature information difficult to access. Furthermore, parsing scientific text and extracting relevant chemical information from text and image data is an open problem of research[18, 19]. While commercial reaction databases exist, e.g., Reaxys[20] or SPRESI[21], the reactions in these databases are mostly unbalanced, not atom-mapped, and lack mechanistic detail[22]. This is in addition to suffering from a severe lack of openness; the databases are exorbitantly priced or provided with a restrictive query interface which precludes serious statistical data mining. As a result, and to the best of our knowledge, effective machine learning approaches to reaction prediction still need to be developed.

## 1.2   A new approach

The limitations of previous work motivate a new, fresh approach to reaction prediction combining machine learning with mechanistic representations. The key idea is to first enumerate all potential source and sink MOs, and thus all possible reactions by their pairing, and then use classification and ranking techniques to identify productive reactions. There are multiple benefits resulting from such an approach. By using very general rules to enumerate possible reactions, the approach is not restricted to manually curated reaction patterns. By detailing individual reactions at the mechanistic level, the system may be able to statistically learn efficient predictive models based on physico-chemical attributes rather than abstract overall transformations. And by ranking possible reactions instead of making binary decisions, the system may provide results amenable to flexible interpretation. However, the new approach also faces three key challenges: (1) the development of appropriate training datasets of productive reactions; (2) the development of a machine learning approach to control the combinatorial complexity resulting from considering all possible pairs of electron donors and acceptors among the reacting molecules; and (3) the development of machine learning solutions to the problem of predictively ranking the possible mechanisms. These challenges are addressed one-by-one in the following sections.

## 2 The data challenge

A mechanistically defined dataset of reactions to use with the proposed approach does not currently exist. To derive a dataset, we use a mechanistically defined rule-based expert system (Reaction Explorer) together with its validation suite[11]. The validation suite is a manually composed set of reactants, reagents, and products covering a complete undergraduate organic chemistry curriculum.

Entering a set of reactants and a reagent model into Reaction Explorer yields the complete sequence of mechanistic steps leading to the final products, where all reactions in this sequence share the conditions encoded by the corresponding reagent model. Each one of these mechanistic steps is considered to be a distinct productive elementary reaction. For a given set of reactants and conditions, which we call a $(r, c)$ query tuple, the Reaction Explorer system labels a small set of reactions productive, while all other reactions enumerated by pairing source and sink MOs over the reactants are considered non-productive.

We then define two $\{0, 1\}$ labels for each atom (up to symmetries) and conditions $(a, c)$ tuple over all $(r, c)$ queries. An $(a, c)$ tuple has label `srcreact = 1` if it is the main atom of a source MO in a productive reaction over any corresponding $(r, c)$ query, and has label `srcreact = 0` otherwise. The label `sinkreact` is defined similarly using sink MOs.

Reaction conditions are described with three parameters: temperature, anion solvation potential, and cation solvation potential. Temperature is listed in Kelvin. The solvation potentials are unitless numbers between 0 and 1 representing ease of cation or anion solvation, thus providing a quantitative scale to describe polar protic, polar aprotic, and nonpolar solvents. Note that any mechanistic interaction with the solvent or reagent is explicitly modeled, e.g. as in Figure 1.

As an initial validation of the method, we consider general ionic reactions from the Reaction Explorer validation suite involving C, H, N, O, Li, Mg, and the halides. Extensions to include stereoselective, pericyclic, and radical reactions are discussed in Section 5. The dataset consists of 6.14 million reactions composed of 84,825 source and 74,725 sink MOs from 2,752 distinct reactants and reaction conditions, i.e., $(r, c)$ queries. Of these 6.14 million reactions, the Reaction Explorer system labels 2,989 of them as productive. There are 22,894 atom symmetry classes, which when paired with reaction condition yields 29,104 $(a, c)$ tuples. Of these 29,104 $(a, c)$ tuples, 1,262 have label `srcreact = 1` , and 1,786 have label `sinkreact = 1`.

Atom and MO interaction data is available at our chemoinformatics portal (`http://cdb.ics.uci.edu`) under Supplements.

## 3 The combinatorial complexity challenge

In the dataset, the average molecule has 44 source MOs and 50 sink MOs. For this average molecule, considering only intermolecular reactions with a second copy of the same molecule gives $44 \times 50 = 2200$ potential elementary reactions. Thus, the number of possible reactions is very large, motivating identifying productive reactions given a $(r, c)$ query in two stages. In the first stage, we train filters using classification techniques on the source and sink reactivity labels. The idea is to train highly sensitive classifiers which reduce the breadth of possible reactions without erroneously filtering productive reactions. Then only those source and sink MOs where the main atom passes the respective atom level filter are considered when enumerating reactions to consider in the second ranking stage for predicting reaction productivity.

Here, we train two separate classifiers to predict the source and sink atom level reactivity labels, each using the same feature descriptions and machine learning implementations. To assess the performance of the reactive site filter training, we perform full 10-fold cross-validation (CV) over all distinct tuples of molecules and conditions $(m, c)$.

### 3.1 Feature representation

Each $(a, c)$ tuple is represented as a vector of physicochemical and topological features. There are 14 real-valued physicochemical features such as the reaction conditions, the molecular weight of the molecule, and the charge at and around the atom. Topological features are meant to capture the neighboring context of $a$ in the molecular graph, for example counts over vertex-and-edge labeled

paths and trees rooted at $a$. We compute paths to length $4$ and trees to depth $2$, producing $743$ molecular graph features. In addition to standard molecular graph features, we also include similar topological features over a restricted alphabet pharmacophore point graph, where pharmacophore point graph definitions are adapted from Hähnke, et al[23]. Using paths of length $4$ and trees of depth $2$ in the pharmacophore point graph yields another $759$ features. This results in a total of $1,516$ features.

## 3.2   Training

Before training, all features are normalized to $[0, 1]$ using the minimum and maximum values of the training set. We oversample $(a, c)$ tuples with label $1$ to ensure approximately balanced classes. We experimented with a variety of architectures. Here we report the results obtained using artificial neural networks using sigmoidal activation functions, with a single hidden layer and a single output node with a cross-entropy error function. Grid search using internal three-fold CV on a single training set is used to fit the architecture size (converging to 10 hidden nodes) and the L2-regularization (weight decay) parameter shared by all folds of the overall 10-fold CV. Weights are optimized by stochastic gradient descent with per-weight adaptive learning rates[24]. Optimization is stopped after 100 epochs as this is observed to be sufficient for convergence.

As highly sensitive classifiers are desired, the choice of a decision threshold is important. We perform internal three-fold CV on the training set to find decision thresholds yielding a false negative rate of 0 on each respective internal test set. The decision threshold for the overall CV fold is taken as the average of these internal CV fold thresholds.

## 3.3   Results

We report the true negative rate (TNR) and the false negative rate (FNR) for both the source and sink classification problems as well as for the the actual reaction filtering problem, as shown in Table 1. In a CV regime, we are able to filter $94.0\%$ of the $6.14$ million non-productive reactions with less than $0.1\%$ false negatives, effectively reducing the ranking problem imbalance by an order of magnitude with minimal error. Having established excellent filtering results with rigorous CV, we then train classifiers with all available data in order to independently assess the ranking method. The results of these classifiers are shown in the last column of Table 1.

Table 1: Reactive site classification results. Source reactive and sink reactive rows show results on the respective classification problems. The reaction row shows results of using the two atom classifiers for an initial reaction filtering. CV columns indicate results of full 10-fold cross-validation over $(m, c)$ tuples. CV results show the mean and standard deviation over folds. The best TNR column shows results when trained with all available data.

| Problem | CV TNR % (SD) | CV FNR % (SD) | Best TNR % |
|---|---|---|---|
| Source Reactive | 87.7(2.0) | 0.1(0.2) | 92.1 |
| Sink Reactive | 75.6(5.8) | 0.2(0.4) | 85.6 |
| Reaction | 94.0(1.5) | $< 0.1(< 0.1)$ | 97.2 |

# 4   The ranking challenge

We pose the task of identifying the productive reactions as a ranking problem. To assess performance, we perform full 10-fold CV over the $2,752$ distinct $(r, c)$ queries. With the overall filtered set of reactions, there are, on average, $1.1$ productive and $62.5$ non-productive reactions per $(r, c)$ query.

## 4.1   Feature representation

Each reaction is composed of a source and sink MO. The reaction feature vector is the concatenation of the corresponding source and sink atom level feature vectors with some modifications. To keep the size reasonable, only real valued and pharmacophore (path length 3 and tree depth 2) atom level

features are included. 124 features are calculated to describe the net difference between reactants and products, such as counts over bond types, rings, and formal charges. And finally, 450 features describing the forward and inverse reactions are calculated, including atoms and bonds involved and implied transition state geometry. This leads to a total of 1,677 reaction features.

## 4.2 Training

We use a pairwise approach to ranking similar to [25], using two identical shared-weight artificial neural networks linked to a single comparator output node with fixed $\pm 1$ weights. The general architecture is shown in Figure 2. Each shared network receives as an input a potential reaction, i.e. a source-sink pair. Training is performed via back-propagation with weight-sharing.

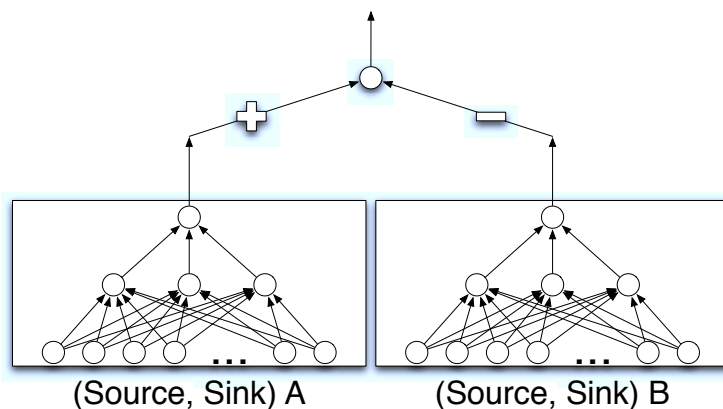

Figure 2: Shared weight artificial neural network architecture for pairwise ranking. The goal is to determine a productivity order between the (source, sink) A and (source, sink) B pairs. This is done with a pair of shared-weight artificial neural networks with sigmoidal hidden nodes and a linear output node. The output of these internal networks are tied to a single sigmoidal output node with fixed weights. The final output will approach 1 if the (source, sink) A pair is predicted to be relatively more productive than the (source, sink) B pair, and 0 otherwise.

Training details are similar to the reactive site classification. All features are normalized to $[0, 1]$ and grid search with internal three-fold CV on a single training set is used to fit the architecture size (converging to 20 hidden nodes) and L2-regularization (weight decay) parameter shared by all folds of the overall 10-fold CV. Weights are optimized using stochastic gradient descent with the same per-weight adaptive learning rate scheme[24]. Optimization is stopped after 25 epochs as this is observed to be sufficient for convergence.

An ensemble consisting of five separate pairwise ranking machines (as described in Figure 2) is used for each training set. Each machine in the ensemble is trained with all the productive reactions (from the training set) and a random partition of the non-productive reactions (from the training set). Final ranking on the test set is determined by either simple majority vote or by ranking the average scores from the linear output node of the inner shared-weight network for each machine in the ensemble. The latter yields a minute performance increase and is reported.

## 4.3 Results

We consider two measures for evaluating rankings, Normalized Discounted Cumulative Gain at list size $i$ (NDCG@$i$) and Percent Within-$n$. NDCG@$i$ is a common information retrieval metric[26] that sums the overall usefulness (or gain) of productive reactions in a given list of the top-$i$ results, where individual gain decays exponentially with lower position. The measure is normalized such that the best possible ranking of a size $i$ list has NDCG@$i = 1$. For example, NDCG@1 is the fraction of $(r, c)$ queries in which the top ranked reaction is a productive reaction. Percent Within-$n$ is simply how many $(r, c)$ queries have at most $n$ non-productive reactions in the smallest ranked list containing all productive reactions. For example, Percent Within-0 measures the percent of $(r, c)$

queries with perfect rank, and Percent Within-4 measures how often all productive reactions are recovered with at most 4 mis-ranked non-productive reactions. Note that the NDCG@1 and Percent Within-0 will differ because roughly $10\%$ of $(r, c)$ queries have more than one productive reaction.

The non-productive MO interactions vastly outnumber the productive interactions. In spite of this imbalance, our approach gives excellent ranking results, shown in Table 2. The NDCG results show, for example, that in $89.5\%$ of the queries, the top ranked reaction is productive. The Percent Within-$n$ results show that $89.1\%$ of queries have perfect ranking, while $99.9\%$ of queries recover all productive reactions by considering lists with at most four non-productive reactions.

Table 2: Reaction ranking results. We show Normalized Discounted Cumulative Gain at different list sizes $i$ (NDCG@$i$) and Percent Within-$n$. See text for description of the measures. We report mean (standard deviation) results over CV folds.

| $i$ | Mean NDCG@$i$ (SD) | $n$ | Percent Within-$n$ (SD) |
|---|---|---|---|
| 1 | 0.895(0.016) | 0 | 89.1(1.7) |
| 2 | 0.939(0.011) | 1 | 96.8(1.0) |
| 3 | 0.952(0.008) | 2 | 98.9(0.6) |
| 4 | 0.954(0.007) | 3 | 99.5(0.4) |
| 5 | 0.956(0.007) | 4 | 99.9(0.3) |

## 4.4   Chemical applications

The strong performance of the ranking system is exhibited by its ability to make accurate multi-step reaction predictions. An example, shown in the first row of Table 3, is an intramolecular Claisen condensation reaction with conditions (room temperature, polar aprotic solvent) requiring three elementary steps. The ranking method correctly predicts the given reaction as the highest ranked reaction at each step.

Table 3: Chemical reactions of interest. The first row shows an example of full multi-step reaction prediction by the ranking system, a three step intramolecular Claisen condensation (room temp., polar aprotic). At each stage, the reaction shown is the top ranked when all possible reactions are considered by the two stage machine learning system. The second row shows two macrocyclizations which the rule-based system (Reaction Explorer) is unable to predict, but the machine learning approach effectively generalizes and ranks correctly. These reactions lead to the formation of a seven homo-cycle (7 carbons) on the left and seven hetero-cycle (6 carbons, 1 oxygen) on the right. The third row shows an intelligible error of the machine learning approach (see text).

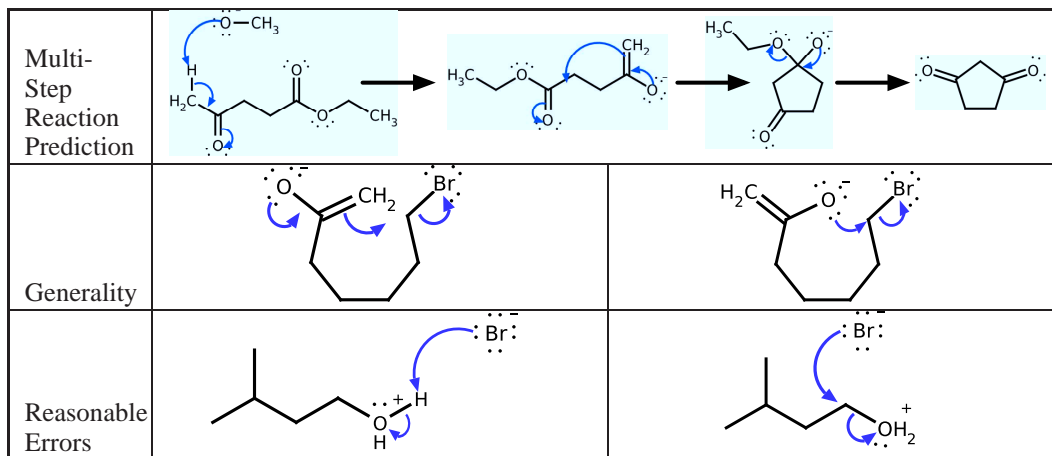

A generalizable system should be able to make reasonable predictions about reactants and reaction types with which it has only had implicit, rather than explicit, experience. Reaction Explorer, as a

rule-based expert system without explicit rules about larger ring forming reactions, does not make any predictions about seven and eight atom cyclizations. In reality though, larger ring forming reactions are possible. The second row of Table 3 shows the top two ranked reactions over a set of bromo-hept-1-en-2-olate reactants, leading to seven-member ring formation. The ranking model, without ever being trained with seven or eight-member ring forming reactions, returns the enolate attack as the most favorable, but also returns the lone pair nucleophilic substitution as the second most favorable. Similar results are made for similar eight-membered ring systems (not shown). Thus the ranking model is able to generalize and make reasonable suggestions, while the rule-based system is limited by hard-coded transformation patterns.

Finally, the vast majority of errors are *close* errors, as exhibited by the 99.9% Within-4 measure. Furthermore, upon examination of these errors, they are largely intelligible and not unreasonable predictions. For example, the third row of Table 3 shows two reactions involving an oxonium compound and a bromide anion. Our ranking models return these two reactions as the highest, ranking the deprotonation slightly ahead of the substitution. This is considered a Within-1 ranking because the Reaction Explorer system labels only the substitution reaction as productive. However, the immediate precursor reaction in the sequence of Reaction Explorer mechanisms leading to these reactants is the inverse of the deprotonation reaction, i.e., the protonation of the alcohol. Hydrogen transfer reactions like this are reversible, and thus the deprotonation is likely the kinetically favored mechanism, i.e., it is reasonable to rank the deprotonation highly. It is just not productive, in that it does not lead to the final overall product. In a prediction system working with multi-step syntheses, such reversals of previous steps are easily discarded.

## 5 Conclusion

Being able to predict the outcome of chemical reactions is a fundamental scientific problem. The ultimate goal of a reaction prediction system is to recapitulate and eventually surpass the ability of human chemists. In this work, we take a significant step in this direction, showing how to formulate reaction prediction as a machine learning problem and building an accurate implementation for a large and key subset of organic chemistry. There are a number of immediate applications of our system, including validating retro-synthetic suggestions, generating virtual libraries of molecules, and mechanistically annotating existing reaction databases.

Reaction prediction is a largely untapped area for machine learning approaches. As such, there is of course room for improvements. The first is increasing the breadth of chemistry captured, e.g. radical, pericyclic, and stereoselective chemistry. Augmenting the MO description with number of electrons, allowing cyclic chained MO interactions, and including face orientations are plausible extensions to attack each of these additional areas of chemical reactivity. A second area of improvement is the curation of larger mechanistically defined datasets. We can approach this manually, by further use of expert systems to construct data with the required level of detail, or by carefully crafted crowd-sourcing approaches. Other ongoing areas of research include improving the features, performing systematic feature selection, and experimenting with different statistical ranking techniques.

As an untapped research problem for the machine learning community, we hope that the current work and our publicly available data will spark continued and open research in this important area.

**Acknowledgments**

Work supported by NIH grants LM010235-01A1 and 5T15LM007743 and NSF grant MRI EIA-0321390 to PB. We acknowledge OpenEye Scientific Software and ChemAxon for academic software licenses. We wish to thank Profs. James Nowick, David Van Vranken, and Gregory Weiss for useful discussions.

**References**

[1] E.J. Corey and W.T. Wipke. Computer-assisted design of complex organic syntheses. *Science*, 166(3902):178–92, 1969.

[2] M.H. Todd. Computer-aided organic synthesis. *Chem. Soc. Rev.*, 34(3):247–266, 2005.

[3] P. Rydberg, D.E. Gloriam, J. Zaretzki, C. Breneman, and L. Olsen. SMARTCyp: A 2D method for prediction of cytochrome P450-mediated drug metabolism. *ACS Med. Chem. Lett.*, 1(3):96–100, 2010.

[4] G. Henkelman, B.P. Uberuaga, and H. Jónsson. A climbing image nudged elastic band method for finding saddle points and minimum energy paths. *J. Chem. Phys.*, 113(22):9901–9904, 2000.

[5] B. Peters, A. Heyden, A.T. Bell, and A. Chakraborty. A growing string method for determining transition states: comparison to the nudged elastic band and string methods. *J. Chem. Phys.*, 120(17):7877–7886, 2004.

[6] C.J. Cramer. *Essentials of Computational Chemistry: Theories and Models*. Wiley, West Sussex, England, 2 edition, 2004.

[7] W.L. Jorgensen, E.R. Laird, A.J. Gushurst, J.M. Fleischer, S.A. Gothe, H.E. Helson, G.D. Paderes, and S. Sinclair. CAMEO: a program from the logical prediction of the products of organic reactions. *Pure Appl. Chem.*, 62:1921–1932, 1990.

[8] R. Hollering, J. Gasteiger, L. Steinhauer, K.-P. Schulz, and A. Herwig. Simulation of organic reactions: from the degradation of chemicals to combinatorial synthesis. *J. Chem. Inf. Model.*, 40(2):482–494, 2000.

[9] G. Benkö, C. Flamm, and P.F. Stadler. A graph-based toy model of chemistry. *J. Chem. Inf. Model.*, 43(4):1085–1093, 2003.

[10] I.M. Socorro, K. Taylor, and J.M. Goodman. ROBIA: a reaction prediction program. *Org. Lett.*, 7(16):3541–3544, 2005.

[11] J. Chen and P. Baldi. No electron left behind: a rule-based expert system to predict chemical reactions and reaction mechanisms. *J. Chem. Inf. Model.*, 49(9):2034–2043, 2009.

[12] P. Röse and J. Gasteiger. Automated derivation of reaction rules for the EROS 6.0 system for reaction prediction. *Anal. Chim. Acta*, 235:163–168, 1990.

[13] B. Wang and Z. Cao. Mechanism of acid-catalyzed hydrolysis of formamide from cluster-continuum model calculations: concerted versus stepwise pathway. *J. Phys. Chem. A*, 114(49):12918–12927, 2010.

[14] C.A. James, D. Weininger, and J. Delany. Daylight theory manual. `http://www.daylight.com/dayhtml/doc/theory/index.html`, 2008. Last accessed January 2011.

[15] C.K. Ingold. *Structure and Mechanism in Organic Chemistry*. Cornell University Press, Ithaca, NY, 1953.

[16] R. Grossman. *The Art of Writing Reasonable Organic Reaction Mechanisms*. Springer-Verlag, New York, NY, 2 edition, 2003.

[17] G. Rozenberg, editor. *Handbook of Graph Grammars and Computing by Graph Transformation: Volume I. Foundations*. World Scientific Publishing, River Edge, NJ, 1997.

[18] D.L. Banville. Mining chemical structural information from the drug literature. *Drug Discovery Today*, 11:35–42, 2006.

[19] J. Park, G.R. Rosania, and K. Saitou. Tunable machine vision-based strategy for automated annotation of chemical databases. *J. Chem. Inf. Model.*, 49(8):1993–2001, 2009.

[20] D.D. Ridley. Searching for chemical reaction information. In S.R. Heller, editor, *The Beilstein Online Database*, volume 436 of *ACS Symposium Series*, pages 88–112. American Chemical Society, Washington, DC, 1990.

[21] D.L. Roth. SPRESIweb 2.1, a selective chemical synthesis and reaction database. *J. Chem. Inf. Model.*, 45(5):1470–1473, 2005.

[22] J. Gasteiger and T. Engel, editors. *Chemoinformatics: A Textbook*. Wiley-VCH, Weinheim, Germany, 2003.

[23] V. Hähnke, B. Hofmann, T. Grgat, E. Proschak, D. Steinhilber, and G. Schneider. PhAST: pharmacophore alignment search tool. *J. Comput. Chem.*, 30(5):761–71, 2009.

[24] R. Neuneier and H.-G. Zimmermann. How to train neural networks. In G.B. Orr and K.-R. Müller, editors, *Neural Networks: Tricks of the Trade*, pages 373–423. Springer-Verlag, Heidelberg, Germany, 1998.

[25] C. Burges, T. Shaked, E. Renshaw, A. Lazier, M. Deeds, N. Hamilton, and G. Hullender. Learning to rank using gradient descent. In *Proceedings of the 22nd International Conference on Machine Learning (ICML05)*, pages 89–96. ACM Press, Bonn, Germany, 2005.

[26] K. Järvelin and J. Kekäläinen. Cumulated gain-based evaluation of IR techniques. *ACM Trans. Inf. Syst.*, 20(4):422–446, 2002.

